# Learning Trajectory and Force Control of an Artificial Muscle Arm by Parallel-hierarchical Neural Network Model

**Masazumi Katayama**　　　**Mitsuo Kawato**
Cognitive Processes Department
ATR Auditory and Visual Perception Research Laboratories
Seika-cho, Soraku-gun, Kyoto 619-02, JAPAN

## Abstract

We propose a new *parallel-hierarchical neural network* model to enable motor learning for simultaneous control of both trajectory and force, by integrating Hogan's control method and our previous neural network control model using a *feedback-error-learning* scheme. Furthermore, two *hierarchical control laws* which apply to the model, are derived by using the Moore-Penrose pseudo-inverse matrix. One is related to the *minimum muscle-tension-change* trajectory and the other is related to the *minimum motor-command-change* trajectory. The human arm is redundant at the dynamics level since joint torque is generated by agonist and antagonist muscles. Therefore, acquisition of the inverse model is an ill-posed problem. However, the combination of these control laws and feedback-error-learning resolve the ill-posed problem. Finally, the efficiency of the parallel-hierarchical neural network model is shown by learning experiments using an artificial muscle arm and computer simulations.

## 1 INTRODUCTION

For humans to properly interact with the environment using their arms, both arm posture and exerted force must be skillfully controlled. The hierarchical neural network model which we previously proposed was successfully applied to trajectory control of an industrial manipulator (Kawato et al., 1987). However, this model could not directly be applied to force control, because the manipulator mechanism was essentially different from the musculo-skeletal system of a human arm. Hogan proposed a biologically motivated control method which specifies both the virtual trajectory and the mechanical impedance of a musculo-skeletal system (Hogan, 1984, 1985). One of its advantages is that both trajectory and force can be simultaneously controlled. However, this control method does not explain motor learning.

In this paper, by integrating these two previous studies, we propose a new *Parallel-Hierarchical Neural network* Model (PHNM) using a *feedback-error-learning* scheme we previously proposed (Kawato et al., 1987), as shown in Fig.1. PHNM explains the biological motor learning for simultaneous control of both trajectory and force. Arm movement depends on the static and dynamic properties of a musculo-skeletal system. From this viewpoint, its inverse model which computes a motor command from a desired trajectory and force, consists of two parallel inverse models: the Inverse Statics Model (ISM) and the Inverse Dynamics Model (ISM) (see Fig.1).

The human arm is redundant at the dynamics level since joint torque is generated by agonist and antagonist muscles. Therefore, acquisition of the inverse model is an ill-posed problem in the sense that the muscle tensions can not be uniquely determined from the prescribed trajectory and force. The central nervous system can resolve the ill-posed problem by applying suitable constraints. Based on behavioral data of human multi-joint arm movement, Uno et al. (1989) found that the trajectory was generated on the criterion that the time integral of the squared sum of the rate of change of muscle tension is minimized. From this point of view, we assume that the central nervous system controls the arm by using two hierarchical objective functions. One objective function is related to the *minimum muscle-tension-change* trajectory. The other objective function is related to the *minimum motor-command-change* trajectory. From this viewpoint, we propose two hierarchical control laws which apply to the feedback controller shown in Fig.1. These control laws are calculated with the Moore-Penrose pseudo-inverse matrix of the Jacobian matrix from muscle tensions or motor commands to joint torque. The combination of these control laws and the feedback-error-learning resolve the ill-posed problem. As a result, the inverse model related to hierarchical objective functions can be acquired by PHNM. We ascertained the efficiency of PHNM by performing experiments in learning control using an artificial-muscle arm with agonist and antagonist muscle-like rubber actuators as shown in Fig.2 (Katayama et al., 1990).

## 2 PARALLEL-HIERARCHICAL NEURAL NETWORK MODEL

In a simple case, the dynamics equation of a human multi-joint arm is described as follows:

$$R(\theta)\ddot{\theta} + B(\theta,\dot{\theta})\dot{\theta} = \tau + G(\theta), \tag{1a}$$

$$\tau = a_f(\theta)T_f(M_f,\theta,\dot{\theta}) - a_e(\theta)T_e(M_e,\theta,\dot{\theta}). \tag{1b}$$

Here, $R(\theta)$ is the inertia matrix, $B(\theta,\dot{\theta})$ expresses a matrix of centrifugal, coriolis and friction forces and $G(\theta)$ is the vector of joint torque due to gravity. $M_f$ and $M_e$ are agonist and antagonist motor commands, $T_f$ and $T_e$ are agonist and antagonist muscle tensions, $\theta$ is the joint-angle, $\tau$ is joint torque generated from the tensions of a pair of muscles and $a_f(\theta)$ and $a_e(\theta)$ are moment arms.

If the arm is static ($\dot{\theta} = \ddot{\theta} = 0$), (1a) and (1b) are reduced to the following:

$$0 = a_f(\theta)T_f(M_f,\theta,0) - a_e(\theta)T_e(M_e,\theta,0) + G(\theta). \tag{2}$$

Therefore, (2) is a statics equation. The problem, which calculates the motor commands from joint angles based on (2), is called the *inverse statics*. There are two difficulties: first, (2) including nonlinear functions ($a_f$, $a_e$, $T_f$, $T_e$ and $G$), must be solved. Second, the inverse statics is an ill-posed problem as mentioned above. These difficulties are resolved by the ISM. The problem of computing dynamic torque other than (2) is called

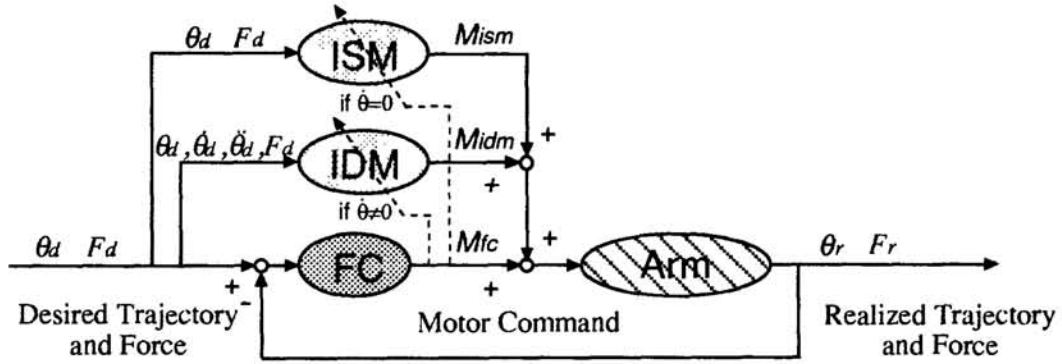

Figure 1: Parallel-Hierarchical Neural Network Model

the *inverse dynamics* and it is resolved by the IDM. The main role of the ISM is to control the equilibrium posture and mechanical stiffness (Hogan, 1984), and that of the IDM is to compensate for dynamic properties of the arm in fast movements. PHNM, in addition to a feedback controller, hierarchically arranges these parallel inverse models. The motor command is the sum of three outputs ($M_{ism}$, $M_{idm}$ and $M_{fc}$) calculated by the ISM, the IDM and the feedback controller, respectively, as shown in Fig.1. The outputs from the ISM and IDM are calculated by feedforward neural networks with synaptic weights $w$ from desired trajectory $\theta_d$ and desired force $F_d$. These neural network models can be described as the mapping from inputs $\theta_d$ and $F_d$ to motor commands. In order to acquire the parallel inverse model, synaptic weights change according to the following *feedback-error-learning* algorithm.

$$\frac{dw}{dt} = \left(\frac{\partial \Psi}{\partial w}\right)^t M_{fc}$$

(3)

The ISM learns when the arm is static and the IDM learns when it is moving. The feedback motor command $M_{fc}$ is fed only to the ISM when $\dot{\theta} = 0$ and only to the IDM when $\dot{\theta} \neq 0$ as an error signal for synaptic modification. The arm is mainly controlled by the feedback controller before learning, and the feedforward control is basically performed only by the parallel inverse model after learning because the output $M_{fc}$ of the feedback controller is minimized after learning. Two control laws which apply to the feedback controller, are derived below.

## 3  HIERARCHICAL CONTROL MECHANISM

In order to acquire the parallel inverse models related to hierarchical objective functions, we propose two control laws reducing the redundancy at the dynamics level, which apply to a feedback controller in the PHNM.

### 3.1  MATHEMATICAL MUSCLE MODEL

Tensions ($T_f$, $T_e$) of agonist and antagonist muscles are generally modeled as follows:

$$T_f = K(M_f)\{\theta_{0,f}(M_f) - \theta\} - B(M_f)\dot{\theta},$$

(4a)

$$T_e = K(M_e)\{\theta - \theta_{0,e}(M_e)\} + B(M_e)\dot{\theta}.$$

(4b)

Here, $M$ consists of $M_f$ and $M_e$ for agonist and antagonist muscles, respectively. The mechanical impedance of a human arm can be manipulated by the stiffness $K(M)$ and viscosity $B(M)$ of the muscle itself, depending on their motor commands. $\theta_{0,f}(M_f)$ and $\theta_{0,e}(M_e)$ are joint angles at equilibrium position. $K(M)$, $B(M)$, $\theta_{0,f}(M_f)$ and $\theta_{0,e}(M_e)$ are approximately given as $K(M) \cong k_0 + kM$, $B(M) \cong b_0 + bM$, $\theta_{0,f}(M_f) \cong \theta_0 + cM_f$ and $\theta_{0,e}(M_e) \cong -\theta_0 - cM_e$, respectively. $k$ and $b$ are coefficients which, respectively, determine elasticity and viscosity. $k_0$ and $b_0$ are intrinsic elasticity and viscosity, respectively. $\theta_0$ is the intrinsic equilibrium angle and $c$ is a constant. Small changes in joint torque are expressed by using the Jacobian matrix $A$ from small changes in motor command to small changes in joint torque. Therefore, by using the Moore-Penrose pseudo-inverse matrix $A^\#$, small changes in motor command are calculated as follows:

$$\begin{pmatrix} \Delta M_f \\ \Delta M_e \end{pmatrix} = A^\# \Delta \tau = \frac{1}{a_f(\theta)^2 \left( C + g_f \right)^2 + a_e(\theta)^2 \left( C - g_e \right)^2} \begin{pmatrix} a_f(\theta)\left( C + g_f \right) \\ a_e(\theta)\left( C - g_e \right) \end{pmatrix} \Delta \tau$$

$$\because C = -(k\theta + b\dot{\theta}), \qquad g_f = k_0 c + k\theta_0 + 2kcM_f$$

$$A^\# = A^T \left( A A^T \right)^{-1}, \qquad g_e = k_0 c + k\theta_0 + 2kcM_e \qquad (5)$$

## 3.2 HIERARCHICAL CONTROL LAWS

Two feedback control laws are explained below, which apply to the feedback controller shown in Fig.1. Firstly, $\Delta T_f = \Delta M_f$ and $\Delta T_e = \Delta M_e$ are given from (4a) and (4b) by assuming $k=b=0$, $c \neq 0$, $a_f(\theta)=a_e(\theta)=a$ and $g_f=g_e=1$ in the simplest case. The solution $A^\# \Delta \tau$ in which the norm $(\Delta T_f^2 + \Delta T_e^2)^{1/2}$ of vector $\Delta T$ is minimized by using the pseudo-inverse matrix $A^\#$, is selected. Therefore, the control law related to the minimum muscle-tension-change trajectory is derived from (5). Then the feedback control law is acquired by using $\Delta \tau = K_p \left( \theta_d - \theta_r \right) + K_d \left( \dot{\theta}_d - \dot{\theta}_r \right) + K_f (F_d - F_r)$. Here, $K_p$, $K_d$ and $K_f$ are feedback gains. Learning is performed by applying the motor commands calculated by this feedback control law to the learning algorithm of (3). As a result, the inverse model is acquired by the PHNM after learning. Only when $a_f(\theta)=a_e(\theta)=a$ does, the inverse model strictly give the optimal solution based on the minimum muscle-tension-change trajectory during the movement. $a$ is a constant moment arm.

Next, another control law is derived from (5) by assuming $k,b \neq 0$, $c=0$, $a_f(\theta)=a_e(\theta)=a$ and $g_f=g_e=1$ by a similar way. In this case, the control law is related to the minimum motor-command-change trajectory, because the norm $(\Delta M_f^2 + \Delta M_e^2)^{1/2}$ of vector $\Delta M$ is minimized by using the pseudo-inverse matrix $A^\#$. Then the control law explains the behavioral data of rapid arm movement, during which the mechanical impedance is increased by coactivation of agonist and antagonist muscles (Kurauchi et al., 1980). The mechanical impedance of the muscles increases when $C$ increases. Therefore, $C$ explains the coactivation because $C$ increases when the arm moves rapidly. Thus, rapid arm movement can be stably executed by such coactivation. It is noted that the control law directly takes account of the variable stiffness and viscosity of the muscle itself. Learning is performed by the same algorithm above. As a result, the inverse model acquired by the PHNM gives the approximate solution related to the minimum motor-command-change trajectory, because $A^\#$ depends on the joint angle in this case. Furthermore, stiffness and

virtual trajectory are uniquely determined from a mathematical muscle model using the outputs of the trained inverse models.

## 4 EFFICIENCY OF PHNM

The efficiency of the PHNM is shown by the experiment results using two hierarchical control laws.

### 4.1 ARTIFICIAL MUSCLE ARM

The artificial muscle arm used in our experiments is the rubber-actuator-arm (5 degrees of freedom, 16 rubber actuators, made by Bridgestone Co.), as shown in Fig.2, which is a manipulator with agonist and antagonist muscle-like actuators. The actuators are made of rubber and driven by air. In our experiment, the motor command is air-pressure.

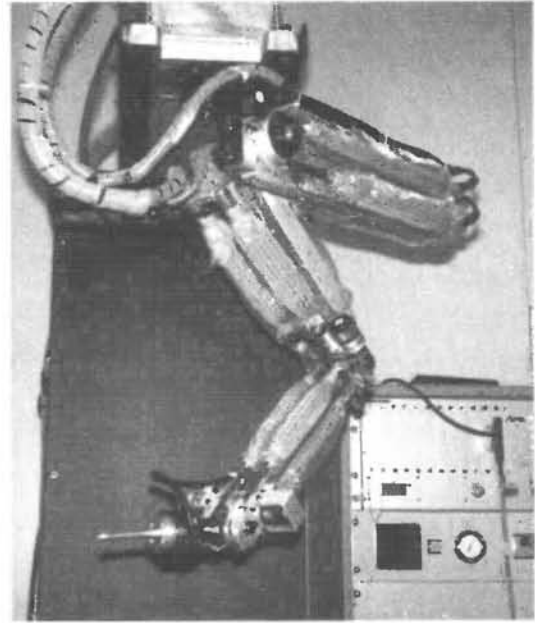

Figure 2: Artificial Muscle Arm

The mechanical structure of the artificial arm is basically the same as that of the human arm. Moreover, properties of the actuator are also similar to those of muscle. The actuator has a variable mechanical impedance which consists of stiffness and viscosity. Then, the stiffness which is mechanically realized, expresses the spring-like behavior of muscle. This property acts as a simple mechanical feedback system whose time delay is "zero". Furthermore, the ratio of the output torque and the weight of the arm is extremely high. Therefore, we hope it will be easy to control the force and trajectory at the end-effector or joint. However, it is difficult to control the trajectory of the arm because the artificial arm, like the human arm, is a very nonlinear system. We note that feedforward control using the trained ISM and IDM is necessary to control the arm.

### 4.2 TRAJECTORY CONTROL OF ARTIFICIAL MUSCLE ARM

Learning control experiments using an artificial muscle arm are performed with the feedback control law related to the *minimum muscle-tension-change* trajectory. The ISM and IDM use a 3-layer perceptron. The results shown in Fig.3 indicate that the conventional feedback control method can not realize accurate trajectory control, because the realized trajectory lagged behind the desired trajectory. While the results shown in Fig.4a indicate that accurate and smooth trajectory control of a slow movement can be realized only by feedforward control using the trained ISM and IDM after learning, because the realized trajectory fits the desired trajectory. Moreover, the result indicates that the PHNM can resolve the ill-posed inverse problem. The results shown in Fig.4b indicate that learning of the ISM and IDM is finished after about 2,000 iterations, because the output of

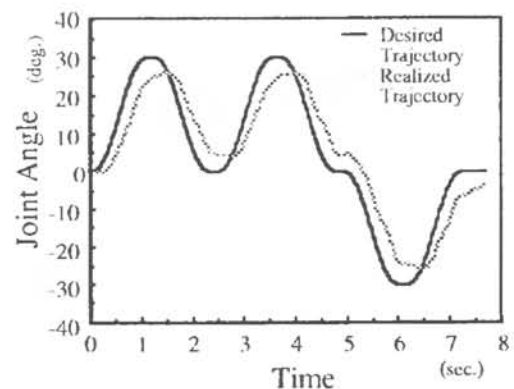

Figure 3: Feedback Control Using Conventional Feedback Controller

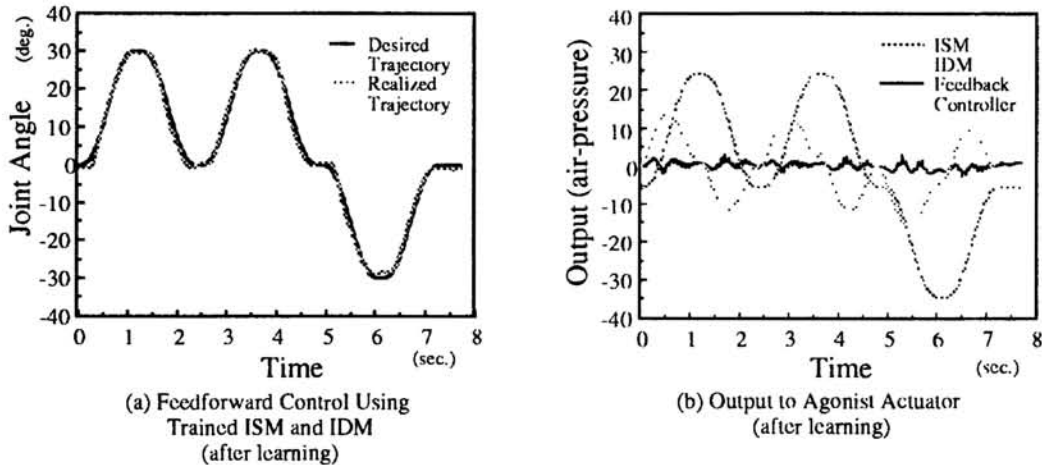

Figure 4:  Trajectory Control Using Control Law
Related to Minimum Muscle-Tension-Change Criterion
(in slow movement using artificial muscle arm)

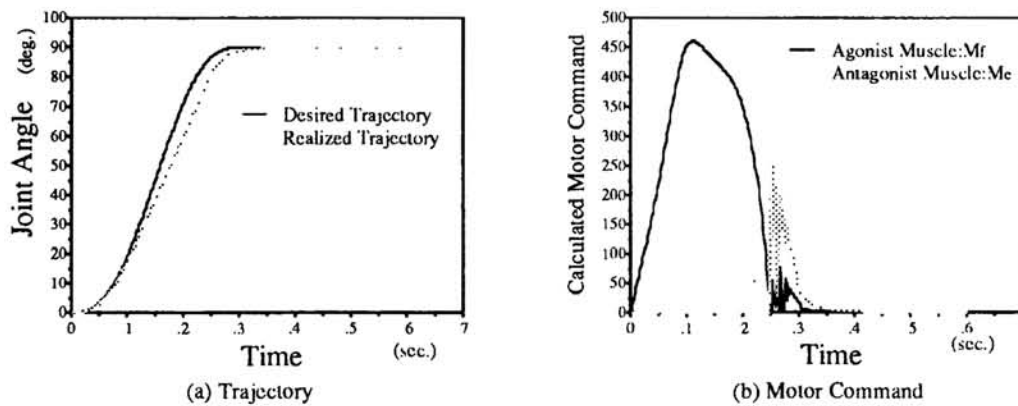

Figure 5:  Feedback Trajectory Control Using Control Law
Related to Minimum Motor-command-change Criterion
(in fast movement using computer simulation)

the feedback controller is minimized.  Then note that the output of the ISM is greater than the other outputs.  Furthermore, we confirmed that by using an untrained trajectory, the generalization capability of the trained parallel inverse models is good.

## 4.3 TRAJECTORY CONTROL IN FAST MOVEMENT

One of the advantages of the control law related to the *minimum motor-command-change* criterion, is shown by a trajectory control experiment in fast movement.  We confirmed that the feedback control law allowed stable trajectory control in fast movement.  Control experiments were performed by computer simulation.  The results shown in Fig.5a indicate that PHNM applying this feedback control law realizes stable trajectory control in rapid movement, because no oscillation characteristics can be found when the arm reaches the desired position.  This is because the mechanical impedance of the joint increases when a pair of muscles are coactivated (see Fig.5b).  Moreover, the results also explain behavioral data in fast arm movement (Kurauchi et al., 1980).

## 4.4 FORCE CONTROL

We confirmed that the feedback control law related to the *minimum motor-command-change* criterion succeeded for accurate force control. The results shown in Fig.6 indicate that accurate force control can be performed by combining the trained IDM and ISM, with PHNM using this feedback control law.

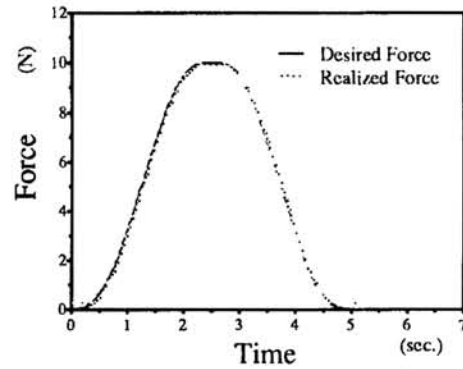

Figure 6: Force Control Using Trained ISM and IDM With Control Law Relate to Minimum Motor-command-change Criterion

## 5  DISCUSSION

The ISM we proposed in this paper has two advantages. The first is that it is easy to train the inverse model of the controlled object because the inverse model is separated into the ISM and IDM. The second is that control using the ISM explains Bizzi's experiment results with a deafferented rhesus monkey (Bizzi et al., 1984). Furthermore the control using the ISM relates to Hogan's control method using the virtual trajectory (Hogan, 1984, 1985).

The Parallel-Hierarchical Neural network Model proposed in this paper integrates Hogan's impedance control and our previous model, and hence can explain motor learning for simultaneous control of both trajectory and force. There is an infinite number of possible combinations of mechanical impedance and virtual trajectory that can produce the same torque and force. Thus, the problem of determining the impedance and the virtual trajectory was ill-posed in Hogan's framework. In the present paper, they were uniquely determined from (5).

## References

[1]  Bizzi, E., Accornero, N., Chapple, W. & Hogan, N. (1984) Posture Control and Trajectory Formation During Arm Movement. *The Journal of Neuroscience*, **4**, 11, 2738-2744.

[2]  Hogan, N. (1984) An Organizing Principle for a Class of Voluntary Movements. *The Journal of Neuroscience*, **4**, 11, 2745-2754.

[3]  Hogan, N. (1985) Impedance Control: An Approach to Manipulation Part I II III. *Journal of Dynamic Systems, Measurement, and Control*, **107**, 1-24.

[4]  Katayama, M. & Kawato, M. (1990) Parallel-Hierarchical Neural Network Model for Motor Control of Musculo-Skeletal System. *The Transactions of The Institute of Electronics, Information and Communication Engineers*, **J73-D-II**, 8, 1328-1335. in Japanese.

[5]  Kawato, M., Furukawa, K. & Suzuki, R. (1987) A Hierarchical Neural-Network Model for Control and Learning of Voluntary Movement. *Biological Cybernetics*, **57**, 169-185.

[6]  Kurauchi, S., Mishima, K. & Kurokawa, T. (1980) Characteristics of Rapid Positional Movements of Forearm. *The Japanese Journal of Ergonomics*, **16**, 5, 263-270. in Japanese.

[7]  Uno, Y., Suzuki, R. & Kawato, M. (1989) Minimum Muscle-Tension-Change Model which Reproduces Human Arm Movement. *Proceedings of the 4th Symposium on Biological and Physiological Engineering*, 299-302. in Japanese.
